# Physiologically Based Speech Synthesis

**Makoto Hirayama**
†ATR Human Information Processing Research Laboratories
2-2, Hikaridai, Seika-cho, Soraku-gun, Kyoto 619-02 Japan

**Eric Vatikiotis-Bateson**
‡ATR Auditory and Visual Perception Research Laboratories

**Kiyoshi Honda‡**  **Yasuharu Koike†**  **Mitsuo Kawato†***

## Abstract

This study demonstrates a paradigm for modeling speech production based on neural networks. Using physiological data from speech utterances, a neural network learns the forward dynamics relating motor commands to muscles and the ensuing articulator behavior that allows articulator trajectories to be generated from motor commands constrained by phoneme input strings and global performance parameters. From these movement trajectories, a second neural network generates PARCOR parameters that are then used to synthesize the speech acoustics.

## 1   INTRODUCTION

Our group has attempted to model speech production computationally as a process in which linguistic intentions are realized as speech through a causal succession of patterned behavior. Our aim is to gain insight into the cognitive and neurophysiological mechanisms governing this complex skilled behavior as well as to provide plausible models of speech synthesis and possibly recognition based on the physiology of speech production. It is the use of physiological data (EMG) representing

motor commands to muscles that distinguishes our modeling effort from those of others who use neural networks for articulation-based synthesis and/or inference of the dynamical constraints on speech motor control (Jordan, 1986, Jordan, 1990, Bailly, Laboissiere, and Schwalz, 1992, Saltzman, 1986, Bengio, Houde, and Jordan, 1992). This paper reports two areas in which implementation of the speech production scheme shown in Figure 1 has progressed. Initially, we concentrated on modeling the dynamics underlying articulation so that phoneme strings can specify motor commands to muscles, which then specify phoneme-specific articulator behavior (Hirayama, Vatikiotis-Bateson, Kawato, and Jordan, 1992). A neural network learned the forward dynamics relating motor commands to muscles and the ensuing articulator behavior associated with prosodically intact, but phonemically simplified, reiterant speech utterances. Then, a cascade neural network (Kawato, Maeda, Uno, and Suzuki, 1990) containing the forward dynamics model along with a suitable smoothness criterion (Uno, Kawato, and Suzuki, 1989) was used to produce continuous motor commands from a sequence of discrete articulatory targets corresponding to the phoneme input string. From this sequence of motor commands, appropriate articulator trajectories were then generated.

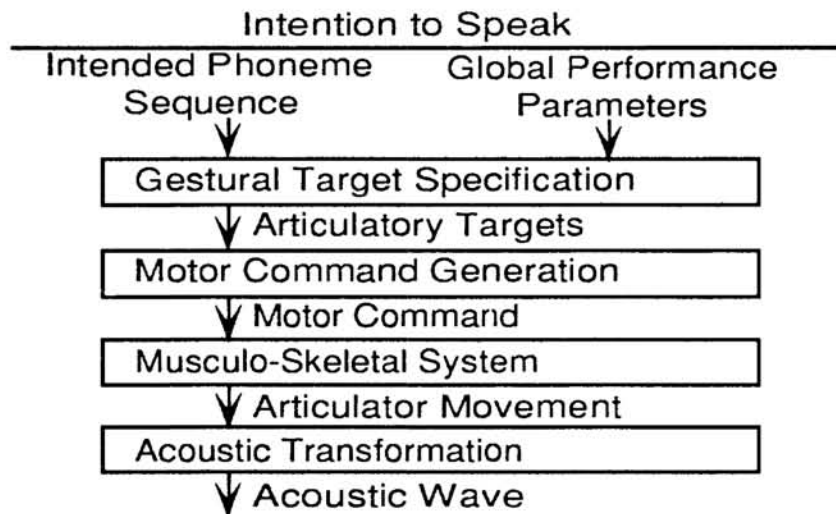

Figure 1: Conceptual scheme of speech production

Although the results of this early work were encouraging, there were two technical limitations obstructing our effort to model real speech. First, using optoelectronic transduction techniques, only simple speech samples whose primary articulators were the lips and jaw could be recorded, hence the use of reiterant *ba*. Without dynamic tongue data, real speech could not be modeled. Also, the reiterant paradigm introduced a degree of rhythmical movement behavior not observed in real speech. The second limitation was that activity of only four muscles and generally only one dimension of articulator motion could be recorded simultaneously. Thus, agonist-antagonist muscle activity was not represented even for this limited set of articulators. Technical improvements in data acquisition and their consequences for the subsequent dynamical modeling of real speech are presented in the next two sections. The second area of progress has been to implement the transform from model- generated articulator trajectories to acoustic output. A neural network is

used to acquire the mapping between articulation and acoustics in terms of PAR-COR parameters (Itakura and Saito, 1969), which are correlated with vocal tract area functions. Speech signals are then generated using a PARCOR synthesizer from articulator input and appropriate glottal sources (currently, the residual of the PARCOR analysis). The results of this modeling for real and reiterant speech are reported in the final section of the paper.

## 2    EMPIRICAL DEVELOPMENTS

In order to acquire data more suitable for real speech modeling, two additional experiments were run in which articulator position, EMG and acoustic data were recorded while the same subject produced real and reiterant speech utterances 5-8 seconds long at different speaking rates and styles (e.g., casual *vs.* precise). In the first of these, a sophisticated optoelectronic device, OPTOTRAK (Northern Digital, Inc.), was used because it permitted simultaneous recording of numerous 3D articulator positions for the lips, jaw and head, ten EMG channels, the speech acoustics, and even dynamic tongue-palate contact patterns. These data were used for modeling of the forward dynamics (see Figure 2) and the forward acoustics. Real speech utterances collected with this system were heavily loaded with labial stops, /p,b,m/, and labiodental fricatives, /f,v/, as well as many low vowels /a, ae/. Since surface EMG was used, it was difficult to obtain reliable recordings of jaw opening (anterior belly of the digastric), and closing (medial pterygoid) muscles. More recently, an electromagnetic position traking system, EMMA (Perkell, Cohen, Svirsky, Matthies, Garabieta, and Jackson, 1992), was used to transduce midsagittal motions of the tongue tip and tongue blade as well as the lips, jaw, and head. Data were collected for the same speech utterances used in the OPTOTRAK and original experiments as well as more natural utterances. Reiterant speech was also recorded for *ta*. For this experiment, surface and hooked-wire EMG techniques were combined, which enabled nine orofacial and extrinsic tongue muscles to be recorded for jaw opening and closing, lip opening and closing, and tongue raising and lowering. The most important aspects of the signal processing for modeling the forward dynamics concern the numerical differentiation of articulator position to obtain velocity and acceleration, and the severe low-pass filtering (including rectification and integration) of the EMG to from 2000 Hz to 20-40 Hz. Both of these introduce spatiotemporal distortions, whose effects on the forward dynamics model are currently being examined.

## 3    MODELING THE FORWARD DYNAMICS

The forward dynamics model was obtained using a 3-layer perceptron with back propagation (Rumelhart, Hinton, and Williams, 1986). Inputs to the network were instantaneous position and velocity for each dimension of articulator motion, and the EMG signals of 9-10 related muscles, which serve as the record of motor commands to muscles; outputs were accelerations for each dimension of motion. Figure 2 shows an example of predicting lip and jaw accelerations from 10 orofacial muscles for the 'natural' test utterance, "Pam put the bobbin in the frying pan and added more puppy parts to the boiling potato soup." As shown by the generalization results in Figure 2, the acquired model produced appropriate acceleration trajectories

for real speech utterances, suggesting that utterance complexity is not a limiting factor in this approach.

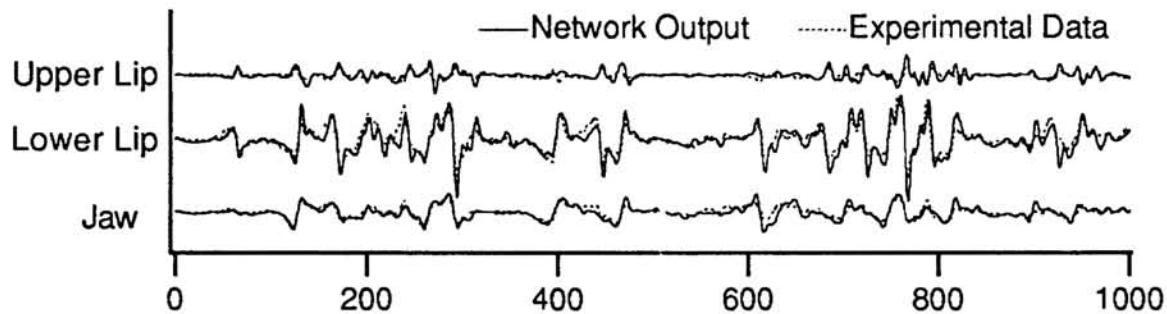

Figure 2: Estimated acceleration over time (5 ms samples) for vertical motion of the three articulators is compared to that of the test sentence: "Pam put the bobbin in the frying pan and added more puppy parts to the boiling potato soup".

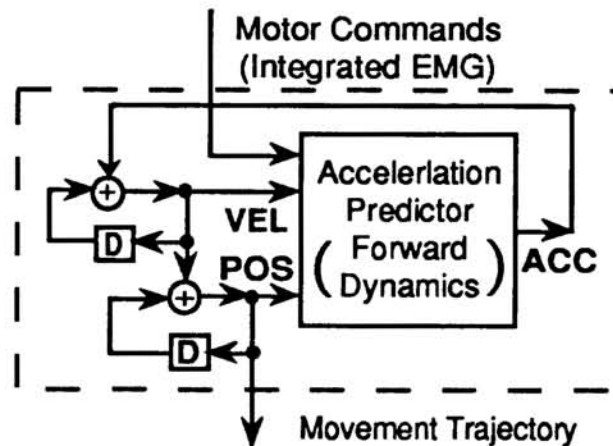

Figure 3: The musculo-skeletal forward dynamics model for producing articulator movement trajectories is implemented as a recurrent network. Continuous motor command (EMG) input drives the network, which uses estimated acceleration at time $tn$, to predict new velocity (integration) and position (double integration) values at the next time step $tn+1$. $D$ is a one-sample delay unit. The network is initialized with position and velocity values taken from the test utterance at $t0$.

Network training resulted in a one-step look-ahead predictor of the articulator dynamics, and was connected recurrently as shown in Figure 3. Using only initial values of articulator position and velocity for the first sample and continuous EMG input, estimated acceleration is looped back and summed with the velocities and positions of the input layer to predict their values for each time step. This is perhaps an overly stringent test of the acquired model because errors are cumulative over the entire utterance 5-8 second utterance. Yet the network outputs appropriate articulator trajectories for the entire utterance. Figure 4 shows the generated trajectory for vertical motion of the jaw during reiterant production of *ba* (recorded with the electromagnetometer). While the trajectory generated by the network tends to underestimate movement amplitude and introduce a small DC offset, it preserves the temporal properties of the test utterance very well everywhere except before a phrasal pause. Although good results have been obtained for the analysis

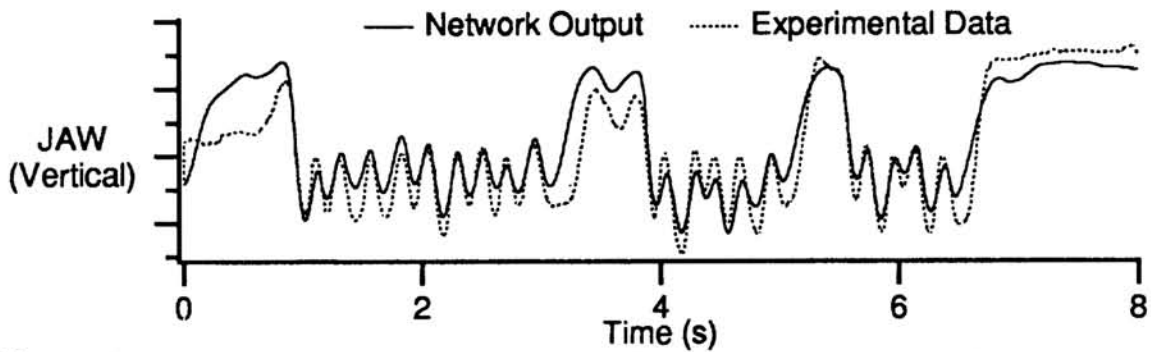

Figure 4: Jaw trajectories, generated by the forward dynamics network are compared with experimental data.

of real speech using the larger sets of articulator and muscle inputs, network complexity has greatly increased. Performance of the full network has been poorer than before in modeling simple reiterant speech, which suggests some form of modularity should be introduced. Also, the addition of tongue data has increased the number of apparent many-to-one mappings between muscle activity and articulator motion. We are now incorporating as a boundary constraint the midsagittal profile of the hard palate and alveolar ridge, against which tongue-tip articulations are made.

# 4    MODELING THE FORWARD ACOUSTICS

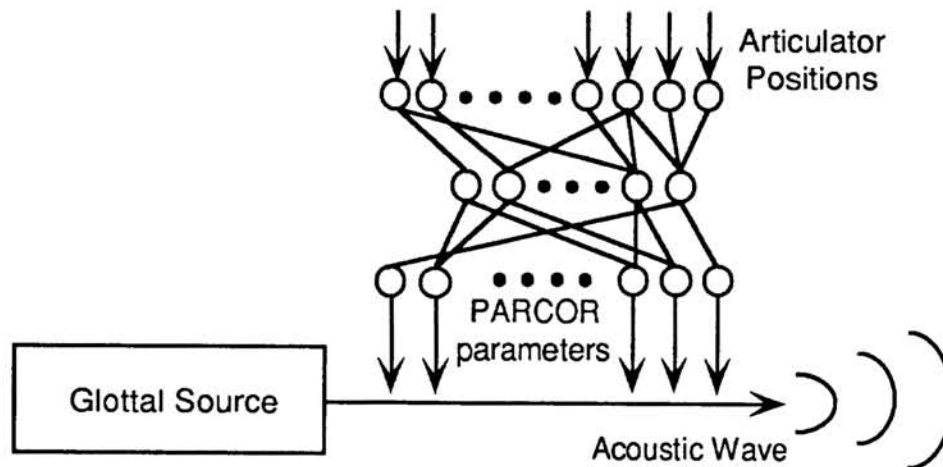

Figure 5: Forward acoustics network.

The final stage of our speech production model entails using a neural network to acquire a model of the relation between articulator motion and the ensuing acoustics. As shown in Figure 5, a 3-layer perceptron, using articulator position as input, was used to learn PARCOR analysis and generate appropriate 16-order PARCOR parameters for subsequent speech synthesis (Itakura and Saito, 1969). We chose PARCOR parameters, rather than more commonly used formant values, because the parameters have some relation to specific cross-sections of the vocal tract – e.g., the first PARCOR corresponds to the cross-sectional area closest to the lips

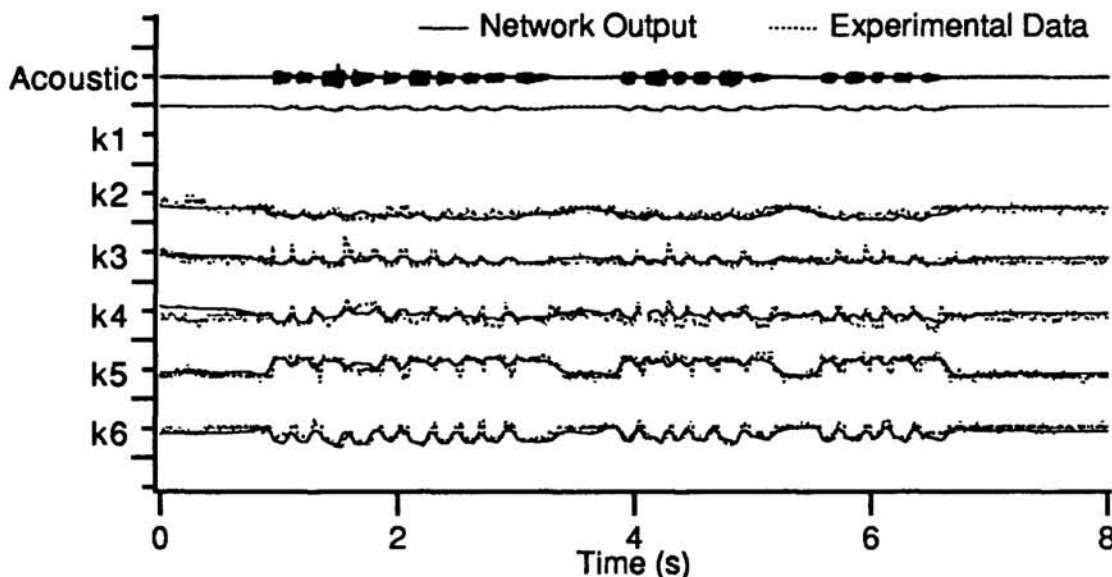

Figure 6: PARCOR parameter values (16-order, 30ms Hanning window at 200Hz) for reiterant *ba* are predicted by the network. Only the first six PARCOR parameters are shown. The range of each parameter is -1 to 1 (small tick beside each wave label indicates 0). The value of *k1* is about 1 during vowels, and network output generally matches the desire wave almost perfectly.

(Wakita, 1973). Also, PARCOR estimation errors do not have the radical consequences that formant estimation errors show. Finally, there is a unique mapping from PARCOR to formant values, but not the reverse (Itakura and Saito, 1969). Figure 6 shows the performance of the PARCOR estimation network for the first 6 parameters out of 16 parameters. Using the learned PARCOR coefficients and a sound source, acoustic signals can be synthesized. Currently, we are investigating various models for controlling sound source as well as prosodic characteristics. However, for this preliminary test of the network's ability to learn PARCOR parameters, the residual signal of PARCOR analysis served as the source waveform. Figure 7 shows an example of the network-learned PARCOR synthesis for reiterant *ba*. In this case, the training result is good as can be seen in the waveform (and frequency spectrum), or by listening the synthesized sound. However, the results have not been as good, so far, for real speech utterances containing a lot of abrupt changes and variability in vocal tract shape. One reason for this may be that learning has not yet converged, because the number articulator input channels is still too limited. So far, we have only two markers on tongue, which is not enough to recover the full vocal tract shape. This situation, hopefully, will improve as data for more tongue positions, or perhaps more functionally motivated placements, are collected. Another reason may be the inherent weakness of PARCOR analysis for modeling dynamic changes in vocal tract shape.

## 5  SUMMARY

This paper outlines two areas of progress in our effort to develop a computational model of speech production. First, we extended our data acquisition to include more

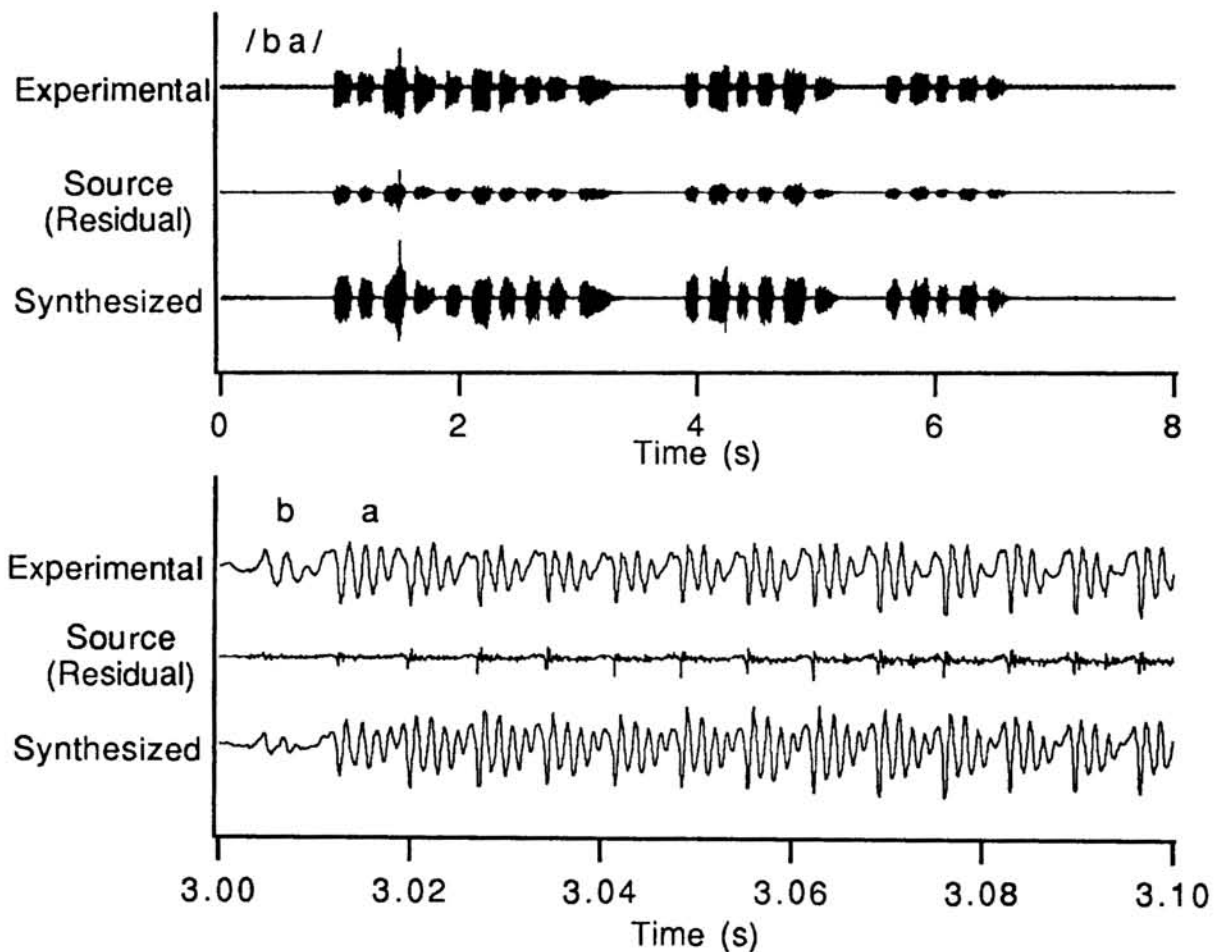

Figure 7: Speech acoustics are synthesized by driving network-learned PARCOR parameters with a glottal source (the residual). The test sentence is reiterant speech using *ba*. Top and bottom graphs differ only in time scale.

muscles and dimensions of motion for more articulators, especially the tongue, so that we could begin modeling the articulatory dynamics of real speech. As hoped, increasing the scope of the data demonstrated the applicability of our network approach to real speech. However, this also increases the size of the network, which has introduced some interesting problems for modeling simple speech samples. We are now considering modifications to the network architecture that will enable adaptive modeling of speech samples, whose complexity (e.g., number of physiological/articulatory components) may vary. Second, we have employed a simple neural network for modeling the articulatory-to-acoustic transform based on PARCOR analysis, whose parameters are correlated with vocal tract shape. Although PARCOR can be used to synthesize speech, its main use for us is as a tool for assessing empirical issues associated with articulatory-acoustic interface.

## Acknowledgments

We thank Haskins Laboratories for use of their facilities (NIH grant DC-00121), Vincent Gracco and Kiyoshi Ohsima for muscle insertions, M. I. Jordan for insightful

discussion, and Yoh'ichi Toh'kura for continuous encouragement. Further support was provided by HFSP grants to M. Kawato.

## Footnotes

*Also, Laboratory of Parallel Distributed Processing, Research Institute for Electronic Science, Hokkaido University, Sapporo, Hokkaido 060, Japan

# References

[1] Bailly, G., Laboissiere, R. and Schwarz, J. L. (1992) Formant trajectories as audible gestures: an alternative for speech synthesis. *Journal of Phonetics*, **19**, 9-23.

[2] Bengio, Y., Houde, J., and Jordan, M. I. (1992) Representations based on articulatory dynamics for speech recognition. Presented at *Neural Networks for Computing*, Snowbird, Utah.

[3] Hirayama, M., Vatikiotis-Bateson, E., Kawato, M., and Jordan, M. I. (1992) Forward dynamics modeling of speech motor control using physiological data. In Moody, J. E., Hanson, S. J., and Lippmann, R. P. (eds.) *Advances in neural information processing systems 4*. San Mateo, CA: Morgan Kaufmann Publishers.

[4] Itakura, F., and Saito, S. (1969) Speech analysis and synthesis by partial correlation parameters. *Proceeding of Japan Acoust. Soc.*, **2-2-6**.

[5] Jordan, M. I. (1986) Serial order: a parallel distributed processing approach. *ICS Report*, **8604**.

[6] Jordan, M. I. (1990) Motor learning and the degrees of freedom problem. In M. Jeannerod (ed.) *Attention and performance XIII*, 796–836, Hillsdale, NJ: Erlbaum.

[7] Kawato, M., Maeda, M., Uno, Y., and Suzuki, R. (1990). Trajectory formation of arm movement by cascade neural-network model based on minimum torque-change criterion. *Biol. Cybern.*, **62**, 275–288.

[8] Perkell, J., Cohen, M., Svirsky, M., Matthies, M., Garabieta, I., and Jackson, M., Electromagnetic midsagittal articulometer systems for transducing speech articulatory movements. *J. Acoust. Soc. Am.*, **92**, 3078–3096.

[9] Rumelhart, D. E., Hinton, G. E., and Williams, R. J. (1986) Learning representations by back-propagating errors. *Nature*, **323**, 533–536.

[10] Saltzman, E. L. (1986) Task dynamic coordination of the speech articulators: A preliminary model. In H. Heuer and C. Fromm (eds.) *Generation and modulation of action patterns*, Berlin: Springer-Verlag.

[11] Uno, Y., Kawato, M., and Suzuki, R. (1989) Formation and control of optimal trajectory in human multijoint arm movement – minimum torque-change model. *Biol. Cybern.*, **61**, 89–101.

[12] Wakita, H. (1973) Direct estimation of the vocal tract shape by inverse filtering of acoustic speech waveforms. *IEEE Trans. Audio Electroacoust.*, **AU-21** 417–427.